# A Generalization of Principal Component Analysis to the Exponential Family

**Michael Collins**       **Sanjoy Dasgupta**       **Robert E. Schapire**

AT&T Labs — Research

180 Park Avenue, Florham Park, NJ  07932

{mcollins, dasgupta, schapire}@research.att.com

## Abstract

Principal component analysis (PCA) is a commonly applied technique for dimensionality reduction. PCA implicitly minimizes a squared loss function, which may be inappropriate for data that is not real-valued, such as binary-valued data. This paper draws on ideas from the Exponential family, Generalized linear models, and Bregman distances, to give a generalization of PCA to loss functions that we argue are better suited to other data types. We describe algorithms for minimizing the loss functions, and give examples on simulated data.

## 1   Introduction

Principal component analysis (PCA) is a hugely popular dimensionality reduction technique that attempts to find a low-dimensional subspace passing close to a given set of points $\mathbf{x}_1, \ldots, \mathbf{x}_n \in \mathbb{R}^d$. More specifically, in PCA, we find a lower dimensional subspace that minimizes the sum of the squared distances from the data points $\mathbf{x}_i$ to their projections $\boldsymbol{\theta}_i$ in the subspace, i.e.,

$$\sum_{i=1}^{n} ||\mathbf{x}_i - \boldsymbol{\theta}_i||^2. \tag{1}$$

This turns out to be equivalent to choosing a subspace that maximizes the sum of the squared lengths of the projections $\boldsymbol{\theta}_i$, which is the same as the (empirical) variance of these projections if the data happens to be centered at the origin (so that $\sum_i \mathbf{x}_i = \mathbf{0}$).

PCA also has another convenient interpretation that is perhaps less well known. In this probabilistic interpretation, each point $\mathbf{x}_i$ is thought of as a random draw from some unknown distribution $P_{\boldsymbol{\theta}_i}$, where $P_{\boldsymbol{\theta}}$ denotes a unit Gaussian with mean $\boldsymbol{\theta} \in \mathbb{R}^d$. The purpose then of PCA is to find the set of parameters $\boldsymbol{\theta}_1, \ldots, \boldsymbol{\theta}_n$ that maximizes the likelihood of the data, subject to the condition that these parameters all lie in a low-dimensional subspace. In other words, $\mathbf{x}_1, \ldots, \mathbf{x}_n$ are considered to be noise-corrupted versions of some true points $\boldsymbol{\theta}_1, \ldots, \boldsymbol{\theta}_n$ which lie in a subspace; the goal is to find these true points, and the main assumption is that the noise is Gaussian. The equivalence of this interpretation to the ones given above follows simply from the fact that negative log likelihood under this Guassian model is equal (ignoring constants) to Eq. (1).

This Gaussian assumption may be inappropriate, for instance if data is binary-valued, or integer-valued, or is nonnegative. In fact, the Gaussian is only one of the canonical distributions that make up the exponential family, and it is a distribution tailored to real-valued

data. The Poisson is better suited to integer data, and the Bernoulli to binary data. It seems natural to consider variants of PCA which are founded upon these other distributions in place of the Gaussian.

We extend PCA to the rest of the exponential family. Let $\{P_{\boldsymbol{\theta}}\}$ be any parameterized set of distributions from the exponential family, where $\boldsymbol{\theta}$ is the natural parameter of a distribution. For instance, a one-dimensional Poisson distribution can be parameterized by $\theta \in \mathbb{R}$, corresponding to mean $\lambda = e^{\theta}$ and distribution $P_{\theta}(n) = e^{n\theta - e^{\theta}}/n! = e^{-\lambda}\lambda^n/n!, \quad n \in \{0, 1, 2, \ldots, \}$. Given data $\mathbf{x}_1, \ldots, \mathbf{x}_n \in \mathbb{R}^d$, the goal is now to find parameters $\boldsymbol{\theta}_1, \ldots, \boldsymbol{\theta}_n$ which lie in a low-dimensional subspace and for which the log-likelihood $\sum_i \log P_{\boldsymbol{\theta}_i}(\mathbf{x}_i)$ is maximized.

Our unified approach effortlessly permits hybrid dimensionality reduction schemes in which different types of distributions can be used for different attributes of the data. If the data $\mathbf{x}_i$ have a few binary attributes and a few integer-valued attributes, then some coordinates of the corresponding $\boldsymbol{\theta}_i$ can be parameters of binomial distributions while others are parameters of Poisson distributions. (However, for simplicity of presentation, in this abstract we assume all distributions are of the same type.)

The dimensionality reduction schemes for non-Gaussian distributions are substantially different from PCA. For instance, in PCA the parameters $\boldsymbol{\theta}_i$, which are means of Gaussians, lie in a space which coincides with that of the data $\mathbf{x}_i$. This is not the case in general, and therefore, although the parameters $\boldsymbol{\theta}_i$ lie in a linear subspace, they typically correspond to a *nonlinear* surface in the space of the data.

The discrepancy and interaction between the space of parameters $\boldsymbol{\theta}$ and the space of the data $\mathbf{x}$ is a central preoccupation in the study of exponential families, generalized linear models (GLM's), and Bregman distances. Our exposition is inevitably woven around these three intimately related subjects. In particular, we show that the way in which we generalize PCA is exactly analogous to the manner in which regression is generalized by GLM's. In this respect, and in others which will be elucidated later, it differs from other variants of PCA recently proposed by Lee and Seung [7], and by Hofmann [4].

We show that the optimization problem we derive can be solved quite naturally by an algorithm that alternately minimizes over the components of the analysis and their coefficients; thus, the algorithm is reminiscent of Csiszár and Tusnády's alternating minization procedures [2]. In our case, each side of the minimization is a simple convex program that can be interpreted as a projection with respect to a suitable Bregman distance; however, the overall program is not generally convex. In the case of Gaussian distributions, our algorithm coincides exactly with the power method for computing eigenvectors; in this sense it is a generalization of one of the oldest algorithms for PCA. Although omitted for lack of space, we can show that our procedure converges in that any limit point of the computed coefficients is a stationary point of the loss function. Moreover, a slight modification of the optimization criterion guarantees the existence of at least one limit point.

Some comments on notation: All vectors in this paper are row vectors. If $\mathbf{M}$ is a matrix, we denote its $i$'th row by $\mathbf{m}_i$ and its $ij$'th element by $m_{ij}$.

## 2 The Exponential Family, GLM's, and Bregman Distances

### 2.1 The Exponential Family and Generalized Linear Models

In the exponential family of distributions the conditional probability of a value $x$ given parameter value $\theta$ takes the following form:

$$\log P(x|\theta) = \log P_0(x) + x\theta - G(\theta). \tag{2}$$

Here, $\theta$ is the "natural parameter" of the distribution, and can usually take any value in the reals. $G(\theta)$ is a function that ensures that the sum (integral) of $P(x|\theta)$ over the domain of $x$ is 1. From this it follows that $G(\theta) = \log \sum_{x \in \mathcal{X}} P_0(x) e^{x\theta}$.

We use $\mathcal{X}$ to denote the domain of $x$. The sum is replaced by an integral in the continuous case, where $P$ defines a density over $\mathcal{X}$. $P_0$ is a term that depends only on $x$, and can usually be ignored as a constant during estimation. The main difference between different members of the family is the form of $G(\theta)$. We will see that almost all of the concepts of the PCA algorithms in this paper stem directly from the definition of $G$.

A first example is a normal distribution, with mean $\mu$ and unit variance, which has a density that is usually written as $\log P(x|\mu) = -\log\sqrt{2\pi} - (x - \mu)^2/2$. It can be verified that this is a member of the exponential family with $\log P_0(x) = -\log\sqrt{2\pi} - x^2/2$, $\theta = \mu$, and $G(\theta) = \theta^2/2$. Another common case is a Bernoulli distribution for the case of binary outcomes. In this case $\mathcal{X} = \{0, 1\}$. The probability of $x \in \mathcal{X}$ is usually written $P(x|p) = p^x (1-p)^{(1-x)}$ where $p$ is a parameter in $[0, 1]$. This is a member of the exponential family with $P_0(x) = 1$, $\theta = \log\frac{p}{1-p}$, and $G(\theta) = \log(1 + e^\theta)$.

A critical function is the derivative $G'(\theta)$, which we will denote as $g(\theta)$ throughout this paper. By differentiating $G(\theta) = \log \sum_{x \in \mathcal{X}} P_0(x) e^{x\theta}$, it is easily verified that $g(\theta) = E[x|\theta]$, the expectation of $x$ under $P(x|\theta)$. In the normal distribution, $E[x|\theta] = \mu$, and in the Bernoulli case $E[x|\theta] = p$. In the general case, $E[x|\theta]$ is referred to as the "expectation parameter", and $g$ defines a function from the natural parameter values to the expectation parameter values.

Our generalization of PCA is analogous to the manner in which generalized linear models (GLM's) [8] provide a unified treatment of regression for the exponential family by generalizing least-squares regression to loss functions that are more appropriate for other members of this family. The regression set-up assumes a training sample of $(\mathbf{x}_i, y_i)$ pairs, where $\mathbf{x}_i \in \mathbb{R}^d$ is a vector of attributes, and $y_i \in \mathbb{R}$ is some response variable. The parameters of the model are a vector $\boldsymbol{\beta} \in \mathbb{R}^d$. The dot product $\boldsymbol{\beta} \cdot \mathbf{x}_i$ is taken to be an approximation of $y_i$. In least squares regression the optimal parameters $\boldsymbol{\beta}^*$ are set to be $\boldsymbol{\beta}^* = \arg\min_{\boldsymbol{\beta} \in \mathbb{R}^d} \sum_i (y_i - \boldsymbol{\beta} \cdot \mathbf{x}_i)^2$.

In GLM's, $h(\boldsymbol{\beta} \cdot \mathbf{x}_i)$ is taken to approximate the expectation parameter of the exponential model, where $h$ is the inverse of the "link function" [8]. A natural choice is to use the "canonical link", where $h = g$, $g$ being the derivative $G'(\theta)$. In this case the natural parameters are directly approximated by $\boldsymbol{\beta} \cdot \mathbf{x}_i$, and the log-likelihood $\sum_i \log P(y_i|x_i)$ is simply $\sum_i \log P_0(y_i) + y_i \boldsymbol{\beta} \cdot \mathbf{x}_i - G(\boldsymbol{\beta} \cdot \mathbf{x}_i)$. In the case of a normal distribution with fixed variance, $G(\theta) = \theta^2/2$ and it follows easily that the maximum-likelihood criterion is equivalent to the least squares criterion. Another interesting case is logistic regression where $G(\theta) = \log(1 + e^\theta)$, and the negative log-likelihood for parameters $\boldsymbol{\beta}$ is $\sum_i \log\left(1 + e^{-y_i^* \boldsymbol{\beta} \cdot \mathbf{x}_i}\right)$ where $y_i^* = 1$ if $y_i = 1$, $y_i^* = -1$ if $y_i = 0$.

## 2.2 Bregman Distances and the Exponential Family

Let $F : \Delta \to \mathbb{R}$ be a differentiable and strictly convex function defined on a closed, convex set $\Delta \subseteq \mathbb{R}$. The Bregman distance associated with $F$ is defined for $p, q \in \Delta$ to be

$$B_F (p \parallel q) \doteq F(p) - F(q) - f(q)(p - q)$$

where $f(x) = F'(x)$. It can be shown that, in general, every Bregman distance is nonnegative and is equal to zero if and only if its two arguments are equal.

For the exponential family the log-likelihood $\log P(x|\theta)$ is directly related to a Bregman

| | normal | Bernoulli | Poisson |
|---|---|---|---|
| $\mathcal{X}$ | $\mathbb{R}$ | $\{0,1\}$ | $\{0,1,2\ldots\infty\}$ |
| $G(\theta)$ | $\theta^2/2$ | $\log(1+e^\theta)$ | $e^\theta$ |
| $g(\theta)$ | $\theta$ | $\frac{e^\theta}{(1+e^\theta)}$ | $e^\theta$ |
| $F(x)$ | $x^2/2$ | $x\log(x)+(1-x)\log(1-x)$ | $x\log(x)-x$ |
| $f(x)=g^{-1}(x)$ | $x$ | $\log\frac{x}{1-x}$ | $\log x$ |
| $B_F\left(p\parallel q\right)$ | $(p-q)^2/2$ | $p\log\frac{p}{q}+(1-p)\log\frac{1-p}{1-q}$ | $p\log\frac{p}{q}+q-p$ |
| $B_F\left(x\parallel g(\theta)\right)$ | $(x-\theta)^2/2$ | $\log(1+e^{-x^*\theta})$ where $x^*=2x-1$ | $e^\theta - x\theta + x\log x - x$ |

Table 1: Various functions of interest for three members of the exponential family

distance. Specifically, [1, 3] define a "dual" function $F$ through $G$ and $g$:

$$F(g(\theta)) + G(\theta) = g(\theta)\theta \tag{3}$$

It can be shown under fairly general conditions that $f(x) = g^{-1}(x)$. Application of these identities implies that the negative log-likelihood of a point can be expressed through a Bregman distance [1, 3]:

$$-\log P(x|\theta) = -\log P_0(x) - F(x) + B_F\left(x\parallel g(\theta)\right). \tag{4}$$

In other words, negative log-likelihood can always be written as a Bregman distance plus a term that is constant with respect to $\theta$ and which therefore can be ignored. Table 1 summarizes various functions of interest for examples of the exponential family.

We will find it useful to extend the idea of Bregman distances to divergences between vectors and matrices. If $\mathbf{x}, \mathbf{y}$ are vectors, and $\mathbf{A}, \mathbf{B}$ are matrices, then we overload the notation as $B_F\left(\mathbf{x}\parallel\mathbf{y}\right) = \sum_i B_F\left(x_i\parallel y_i\right)$ and $B_F\left(\mathbf{A}\parallel\mathbf{B}\right) = \sum_i\sum_j B_F\left(a_{ij}\parallel b_{ij}\right)$. (The notion of Bregman distance as well as our generalization of PCA can be extended to vectors in a more general manner; here, for simplicity, we restrict our attention to Bregman distances and PCA problems of this particular form.)

## 3 PCA for the Exponential Family

We now generalize PCA to other members of the exponential family. We wish to find $\boldsymbol{\theta}_i$'s that are "close" to the $\mathbf{x}_i$'s and which belong to a lower dimensional subspace of parameter space. Thus, our approach is to find a basis $\mathbf{v}_1,\ldots,\mathbf{v}_\ell$ in $\mathbb{R}^d$ and to represent each $\boldsymbol{\theta}_i$ as the linear combination of these elements $\boldsymbol{\theta}_i = \sum_k a_{ik}\mathbf{v}_k$ that is "closest" to $\mathbf{x}_i$.

Let $\mathbf{X}$ be the $n \times d$ matrix whose $i$'th row is $\mathbf{x}_i$. Let $\mathbf{V}$ be the $\ell \times d$ matrix whose $k$'th row is $\mathbf{v}_k$, and let $\mathbf{A}$ be the $n \times \ell$ matrix with elements $a_{ik}$. Then $\boldsymbol{\Theta} = \mathbf{AV}$ is an $n \times d$ matrix whose $i$'th row is $\boldsymbol{\theta}_i$ as above. This is a matrix of natural parameter values which define the probability of each point in $\mathbf{X}$.

Following the discussion in Section 2, we consider the loss function taking the form

$$L(\mathbf{V}, \mathbf{A}) = -\log P(\mathbf{X}|\mathbf{A}, \mathbf{V}) = -\sum_i\sum_j \log P(x_{ij}|\theta_{ij}) = C + \sum_i\sum_j(-x_{ij}\theta_{ij}+G(\theta_{ij}))$$

where $C$ is a constant term which will be dropped from here on. The loss function varies depending on which member of the exponential family is taken, which simply changes the form of $G$. For example, if $\mathbf{X}$ is a matrix of real values, and the normal distribution is appropriate for the data, then $G(\theta) = \theta^2/2$ and the loss criterion is the usual squared loss for PCA. For the Bernoulli distribution, $G(\theta) = \log(1 + e^\theta)$. If we define $x_{ij}^* = 2x_{ij} - 1$, then $L(\mathbf{V}, \mathbf{A}) = \sum_i\sum_j \log(1 + e^{-x_{ij}^*\theta_{ij}})$.

From the relationship between log-likelihood and Bregman distances (see Eq. (4)), the loss can also be written as

$$L(\mathbf{V}, \mathbf{A}) = \sum_i \sum_j B_F\left(x_{ij} \;\|\; g(\theta_{ij})\right) = \sum_i B_F\left(\mathbf{x}_i \;\|\; g(\boldsymbol{\theta}_i)\right)$$

(where we allow $g$ to be applied to vectors and matrices in a pointwise manner). Once $\mathbf{V}$ and $\mathbf{A}$ have been found for the data points, the $i$'th data point $\mathbf{x}_i \in \mathbb{R}^d$ can be represented as the vector $\mathbf{a}_i$ in the lower dimensional space $\mathbb{R}^\ell$. Then $\mathbf{a}_i$ are the coefficients which define a Bregman projection of the vector $\mathbf{x}_i$:

$$\mathbf{a}_i = \arg\min_{\mathbf{a} \in \mathbb{R}^\ell} B_F\left(\mathbf{x}_i \;\|\; g(\mathbf{a}\mathbf{V})\right). \tag{5}$$

The generalized form of PCA can also be considered to be search for a low dimensional basis (matrix $\mathbf{V}$) which defines a surface that is close to all the data points $\mathbf{x}_i$. We define the set of points $\mathcal{Q}(\mathbf{V})$ to be $\mathcal{Q}(\mathbf{V}) = \left\{ g(\mathbf{a}\mathbf{V}) \mid \mathbf{a} \in \mathbb{R}^\ell \right\}$. The optimal value for $\mathbf{V}$ then minimizes the sum of projection distances: $\mathbf{V}^* = \arg\min_{\mathbf{V}} \sum_i \min_{\mathbf{q} \in \mathcal{Q}(\mathbf{V})} B_F\left(\mathbf{x}_i \;\|\; \mathbf{q}\right)$. Note that for the normal distribution $g(\theta) = \theta$ and the Bregman distance is Euclidean distance so that the projection operation in Eq. (5) is a simple linear projection ($\mathbf{a}_i = \mathbf{x}_i\mathbf{V}^{\mathrm{T}}$). $\mathcal{Q}(\mathbf{V})$ is also simplified in the normal case, simply being the hyperplane whose basis is $\mathbf{V}$.

To summarize, once a member of the exponential family — and by implication a convex function $G(\theta)$ — is chosen, regular PCA is generalized in the following way:

• The loss function is negative log-likelihood, $-\log P(x|\theta) = -x\theta + G(\theta) + \text{constant}$.

• The matrix $\boldsymbol{\Theta} = \mathbf{A}\mathbf{V}$ is taken to be a matrix of natural parameter values.

• The derivative $g(\theta)$ of $G(\theta)$ defines a matrix of expectation parameters, $g(\mathbf{A}\mathbf{V})$.

• A function $F$ is derived from $G$ and $g$. A Bregman distance $B_F$ is derived from $F$.

• The loss is a sum of Bregman distances from the elements $x_{ij}$ to values $g(\theta_{ij})$.

• PCA can also be thought of as search for a matrix $\mathbf{V}$ that defines a surface $\mathcal{Q}(\mathbf{V})$ which is "close" to all the data points.

The normal distribution is a simple case because $g(\theta) = \theta$, and the divergence is Euclidean distance. The projection operation is a linear operation, and $\mathcal{Q}(\mathbf{V})$ is the hyperplane which has $\mathbf{V}$ as its basis.

## 4  Generic Algorithms for Minimizing the Loss Function

We now describe a generic algorithm for minimization of the loss function. First, we concentrate on the simplest case where there is just a single component so that $\ell = 1$. (We drop the $k$ subscript from $a_{ik}$ and $v_{jk}$.) The method is iterative, with an initial random choice for the value of $\mathbf{V}$. Let $\mathbf{V}^{(t)}, \mathbf{A}^{(t)}$, etc. denote the values at the $t$'th iteration, and let $\mathbf{V}^{(0)}$ be the initial random choice. We propose the iterative updates $\mathbf{A}^{(t)} = \arg\min_{\mathbf{A}} L(\mathbf{V}^{(t-1)}, \mathbf{A})$ and $\mathbf{V}^{(t)} = \arg\min_{\mathbf{V}} L(\mathbf{V}, \mathbf{A}^{(t)})$. Thus $L$ is alternately minimized with respective to its two arguments, each time optimizing one argument while keeping the other one fixed, reminiscent of Csiszár and Tusnády's alternating minization procedures [2].

It is useful to write these minimization problems as follows:

$$\text{For } i = 1 \ldots n, \qquad a_i^{(t)} = \arg\min_{a \in \mathbb{R}} \sum_j B_F\left(x_{ij} \;\|\; g(a v_j^{(t-1)})\right)$$
$$\text{For } j = 1 \ldots d, \qquad v_j^{(t)} = \arg\min_{v \in \mathbb{R}} \sum_i B_F\left(x_{ij} \;\|\; g(a_i^{(t)} v)\right).$$

We can then see that there are $n + d$ optimization problems, and that each one is essentially identical to a GLM regression problem (a very simple one, where there is a single parameter being optimized over). These sub-problems are easily solved, as the functions are convex in the argument being optimized over, and the large literature on maximum-likelihood estimation in GLM's can be directly applied to the problem.

These updates take a simple form for the normal distribution: $\mathbf{A}^{(t)} = \mathbf{X}(\mathbf{V}^{(t-1)})^{\mathrm{T}}/\|\mathbf{V}^{(t-1)}\|^2$, and $\mathbf{V}^{(t)} = (\mathbf{A}^{(t)})^{\mathrm{T}}\mathbf{X}/\|\mathbf{A}^{(t)}\|^2$. It follows that $\mathbf{V}^{(t)} = \mathbf{V}^{(t-1)}\mathbf{X}^{\mathrm{T}}\mathbf{X}/C$, where $C$ is a scalar value. The method is then equivalent to the power method (see Jolliffe [5]) for finding the eigenvector of $\mathbf{X}^{\mathrm{T}}\mathbf{X}$ with the largest eigenvalue, which is the best single component solution for $\mathbf{V}$. Thus the generic algorithm generalizes one of the oldest algorithms for solving the regular PCA problem.

The loss is convex in either of its arguments with the other fixed, but in general is not convex in the two arguments together. This makes it very difficult to prove convergence to the global minimum. The normal distribution is an interesting special case in this respect — the power method is known to converge to the optimal solution, in spite of the non-convex nature of the loss surface. A simple proof of this comes from properties of eigenvectors (Jolliffe [5]). It can also be explained by analysis of the Hessian $\mathbf{H}$: for any stationary point which is not the global minimum, $\mathbf{H}$ is not positive semi-definite. Thus these stationary points are saddle points rather than local minima. The Hessian for the generalized loss function is more complex; it remains an open problem whether it is also not positive semi-definite at stationary points other than the global minimum. It is also open to determine under which conditions this generic algorithm will converge to a global minimum. In preliminary numerical studies, the algorithm seems to be well behaved in this respect. Moreover, any limit point of the sequence $\mathbf{\Theta}^{(t)} = \mathbf{A}^{(t)}\mathbf{V}^{(t)}$ will be a stationary point.

However, it is possible for this sequence to diverge since the optimum may be at infinity. To avoid such degenerate choices of $\mathbf{\Theta}$, we can use a modified loss

$$\sum_i \sum_j \left[ B_F\left( x_{ij} \ \| \ g(\theta_{ij}) \right) + \varepsilon B_F\left( \mu_0 \ \| \ g(\theta_{ij}) \right) \right]$$

where $\varepsilon$ is a small positive constant, and $\mu_0$ is any value in the range of $g$ (and therefore for which $g^{-1}(\mu_0)$ is finite). This is roughly equivalent to adding a conjugate prior and finding the maximum a posteriori solution. It can be proved, for this modified loss, that the sequence $\mathbf{\Theta}^{(t)}$ remains in a bounded region and hence always has at least one limit point which must be a stationary point. (All proofs omitted for lack of space.)

There are various ways to optimize the loss function when there is more than one component. We give one algorithm which cycles through the $\ell$ components, optimizing each in turn while the others are held fixed:

//**Initialization**
   Set $\mathbf{A} = 0$, $\mathbf{V} = 0$
//**Cycle through $\ell$ components $N$ times**
     For $n = 1, \ldots, N$, $c = 1, \ldots, \ell$:
//**Now optimize the $c$'th component with other components fixed**
     Initialize $\mathbf{v}_c^{(0)}$ randomly, and set $s_{ij} = \sum_{k \neq c} a_{ik} v_{kj}$
     For $t = 1, \ldots,$ convergence

       For $i = 1, \ldots, n$,    $a_{ic}^{(t)} = \arg\min_{a \in \mathbb{R}} \sum_j B_F\left( x_{ij} \ \| \ g(a v_{cj}^{(t-1)} + s_{ij}) \right)$

       For $j = 1 \ldots d$,    $v_{cj}^{(t)} = \arg\min_{v \in \mathbb{R}} \sum_i B_F\left( x_{ij} \ \| \ g(a_{ic}^{(t)} v + s_{ij}) \right)$

The modified Bregman projections now include a term $s_{ij}$ representing the contribution of the $\ell - 1$ fixed components. These sub-problems are again a standard optimization problem regarding Bregman distances, where the terms $s_{ij}$ form a "reference prior".

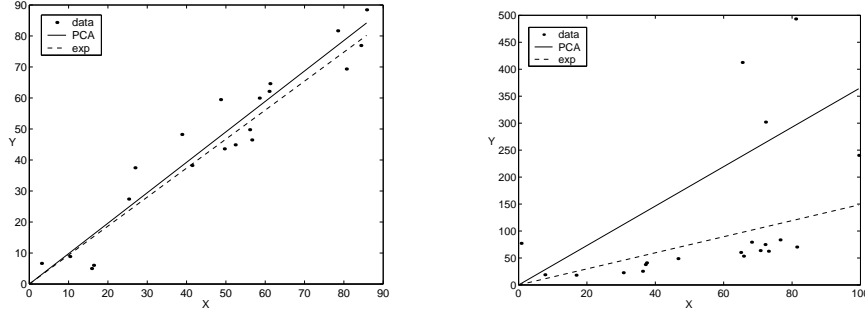

Figure 1: Regular PCA vs. PCA for the exponential distribution.

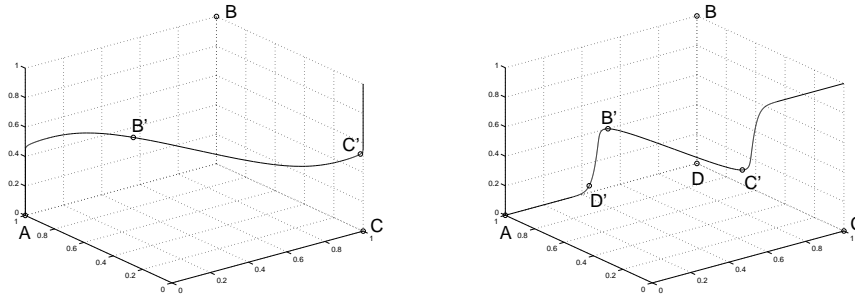

Figure 2: Projecting from 3- to 1-dimensional space, via Bernoulli PCA. Left: the three points $A, B, C$ are projected onto a one-dimensional curve. Right: point $D$ is added.

## 5   Illustrative examples

**Exponential distribution.** Our generalization of PCA behaves rather differently for different members of the exponential family. One interesting example is that of the *exponential* distributions on nonnegative reals. For one-dimensional data, these densities are usually written as $\alpha e^{-\alpha x}$, where $\frac{1}{\alpha}$ is the mean. In the uniform system of notation we have been using, we would instead index each distribution by a single natural parameter $\theta \in (-\infty, 0)$ (basically, $\theta = -\alpha$), and write the density as $P_\theta(x) = e^{\theta x - G(\theta)}$, where $G(\theta) = -\ln(-\theta)$. The link function in this case is $g(\theta) = -\frac{1}{\theta}$, the mean of the distribution.

Suppose we are given data $\mathbf{X} \in \mathbb{R}^{n \times d}$ and want to find the best one-dimensional approximation: a vector $\mathbf{v}$ and coefficients $\mathbf{a}$ such that the approximation $\mathbf{x}_i \approx g(a_i \mathbf{v}), i = 1, \ldots, n$, has minimum loss. The alternating minimization procedure of the previous section has a simple closed form in this case, consisting of the iterative update rule

$$\frac{1}{\mathbf{v}} \longleftarrow \frac{n}{d} \mathbf{X}^T \cdot \frac{1}{\mathbf{X}\mathbf{v}}.$$

Here the shorthand $\frac{1}{\mathbf{v}}$ denotes a componentwise reciprocal, i.e., $(\frac{1}{v_1}, \ldots, \frac{1}{v_d})$. Notice the similarity to the update rule of the power method for PCA: $\mathbf{v} \leftarrow \mathbf{X}^T \mathbf{X}\mathbf{v}$. Once $\mathbf{v}$ is found, we can recover the coefficients $\mathbf{a} = -d \cdot \frac{1}{\mathbf{X}\mathbf{v}}$. The points $\boldsymbol{\theta}_i = a_i \mathbf{v}$ lie on a line through the origin. Normally, we would not expect the points $g(\boldsymbol{\theta}_i)$ to also lie on a straight line; however, in this case they do, because any point of the form $g(a\mathbf{v}), a \in \mathbb{R}$, can be written as $-\frac{1}{a} \cdot \frac{1}{\mathbf{v}}$ and so must lie in the direction $\frac{1}{\mathbf{v}}$.

Therefore, we can reasonably ask how the lines found under this exponential assumption differ from those found under a Gaussian assumption (that is, those found by regular PCA), provided all data is nonnegative. As a very simple illustration, we conducted two toy experiments with twenty data points in $\mathbb{R}^2$ (Figure 1). In the first, the points all lay very close

to a line, and the two versions of PCA produced similar results. In the second experiment, a few of the points were moved farther afield, and these outliers had a larger effect upon regular PCA than upon its exponential variant.

**Bernoulli distribution.** For the Bernoulli distribution, a linear subspace of the space of parameters is typically a nonlinear surface in the space of the data. In Figure 2 (left), three points in the three-dimensional hypercube $\{0, 1\}^3$ are mapped via our PCA to a one-dimensional curve. The curve passes through one of the points ($A$); the projections of the two other ($B \to B'$ and $C \to C'$) are indicated. Notice that the curve is symmetric about the center of the hypercube, $(1/2, 1/2, 1/2)$. In Figure 2 (right), another point (D) is added, and causes the approximating one-dimensional curve to swerve closer to it.

## 6  Relationship to Previous Work

Lee and Seung [6, 7] and Hofmann [4] also describe probabilistic alternatives to PCA, tailored to data types that are not gaussian. In contrast to our method, [4, 6, 7] approximate *mean* parameters underlying the generation of the data points, with constraints on the matrices $\mathbf{A}$ and $\mathbf{V}$ ensuring that the elements of $\mathbf{AV}$ are in the correct domain. By instead choosing to approximate the natural parameters, in our method the matrices $\mathbf{A}$ and $\mathbf{V}$ do not usually need to be constrained—instead, we rely on the link function $g$ to give a transformed matrix $g(\mathbf{AV})$ which lies in the domain of the data points.

More specifically, Lee and Seung [6] use the loss function $\sum_i \sum_j \left( -x_{ij} \log \theta_{ij} + \theta_{ij} \right)$ (ignoring constant factors, and again defining $\theta_{ij} = \sum_k a_{ik} v_{kj}$). This is optimized with the constraint that $\mathbf{A}$ and $\mathbf{V}$ should be positive. This method has a probabilistic interpretation, where each data point $x_{ij}$ is generated from a Poisson distribution with mean parameter $\theta_{ij}$. For the Poisson distribution, our method uses the loss function $\sum_i \sum_j \left( -x_{ij} \theta_{ij} + e^{\theta_{ij}} \right)$, but without any constraints on the matrices $\mathbf{A}$ and $\mathbf{V}$. The algorithm in Hofmann [4] uses a loss function $\sum_i \sum_j x_{ij} \log \theta_{ij}$, where the matrices $\mathbf{A}$ and $\mathbf{V}$ are constrained such that all the $\theta_{ij}$'s are positive, and also such that $\sum_{i,j} \theta_{ij} = 1$.

Bishop and Tipping [9] describe probabilistic variants of the gaussian case. Tipping [10] discusses a model that is very similar to our case for the Bernoulli family.

**Acknowledgements.** This work builds upon intuitions about exponential families and Bregman distances obtained largely from interactions with Manfred Warmuth, and from his papers. Thanks also to Andreas Buja for several helpful comments.

## References

[1] Katy S. Azoury and M. K. Warmuth. Relative loss bounds for on-line density estimation with the exponential family of distributions. *Machine Learning*, 43:211–246, 2001.

[2] I. Csisz´ar and G. Tusn´ady. Information geometry and alternating minimization procedures. *Statistics and Decisions, Supplement Issue*, 1:205–237, 1984.

[3] Jürgen Forster and Manfred Warmuth. Relative expected instantaneous loss bounds. *Journal of Computer and System Sciences*, to appear.

[4] Thomas Hofmann. Probabilistic latent semantic indexing. In *Proceedings of the 22nd Annual International ACM SIGIR Conference on Research and Development in Information Retrieval*, 1999.

[5] I. T. Jolliffe. *Principal Component Analysis*. Springer-Verlag, 1986.

[6] D. D. Lee and H. S. Seung. Learning the parts of objects with nonnegative matrix factorization. *Nature*, 401:788, 1999.

[7] Daniel D. Lee and H. Sebastian Seung. Algorithms for non-negative matrix factorization. In *Advances in Neural Information Processing Systems 13*, 2001.

[8] P. McCullagh and J. A. Nelder. *Generalized Linear Models*. CRC Press, 2nd edition, 1990.

[9] M. E. Tipping and C. M. Bishop. Probabilistic principal component analysis. *Journal of the Royal Statistical Society, Series B*, 61(3):611–622, 1999.

[10] Michael E. Tipping. Probabilistic visualisation of high-dimensional binary data. In *Advances in Neural Information Processing Systems 11*, pages 592–598, 1999.
